# Sidestepping Intractable Inference
# with Structured Ensemble Cascades

**David Weiss**[*]     **Benjamin Sapp**[*]     **Ben Taskar**
Computer and Information Science
University of Pennsylvania
Philadelphia, PA 19104, USA
`{djweiss,bensapp,taskar}@cis.upenn.edu`

## Abstract

For many structured prediction problems, complex models often require adopting approximate inference techniques such as variational methods or sampling, which generally provide no satisfactory accuracy guarantees. In this work, we propose sidestepping intractable inference altogether by learning ensembles of tractable sub-models as part of a structured prediction cascade. We focus in particular on problems with high-treewidth and large state-spaces, which occur in many computer vision tasks. Unlike other variational methods, our ensembles do not enforce agreement between sub-models, but filter the space of possible outputs by simply adding and thresholding the max-marginals of each constituent model. Our framework jointly estimates parameters for all models in the ensemble for each level of the cascade by minimizing a novel, convex loss function, yet requires only a linear increase in computation over learning or inference in a single tractable sub-model. We provide a generalization bound on the filtering loss of the ensemble as a theoretical justification of our approach, and we evaluate our method on both synthetic data and the task of estimating articulated human pose from challenging videos. We find that our approach significantly outperforms loopy belief propagation on the synthetic data and a state-of-the-art model on the pose estimation/tracking problem.

## 1   Introduction

We address the problem of prediction in graphical models that are computationally challenging because of both high-treewidth and large state-spaces. A primary example where intractable, large state-space models typically arise is in dynamic state estimation problems, including tracking articulated objects or multiple targets [1, 2]. The complexity stems from interactions of multiple degrees-of-freedom (state variables) and fine-level resolution at which states need to be estimated. Another typical example arises in pixel-labeling problems where the model topology is typically a 2D grid and the number of classes is large [3]. In this work, we propose a novel, principled framework called Structured Ensemble Cascades for handling state complexity while learning complex models, extending our previous work on structured cascades for low-treewidth models [4].

The basic idea of structured cascades is to learn a sequence of coarse-to-fine models that are optimized to safely filter and refine the structured output state space, speeding up both learning and inference. While we previously assumed (sparse) exact inference is possible throughout the cascade [4], in this work, we apply and extend the structured cascade framework to *intractable* high-treewidth models. To avoid intractable inference, we decompose the desired model into an ensemble of tractable sub-models for each level of the cascade. For example, in the problem of tracking articulated human pose, each sub-model includes temporal dependency for a single body joint only.

---

[*]These authors have contributed equally.

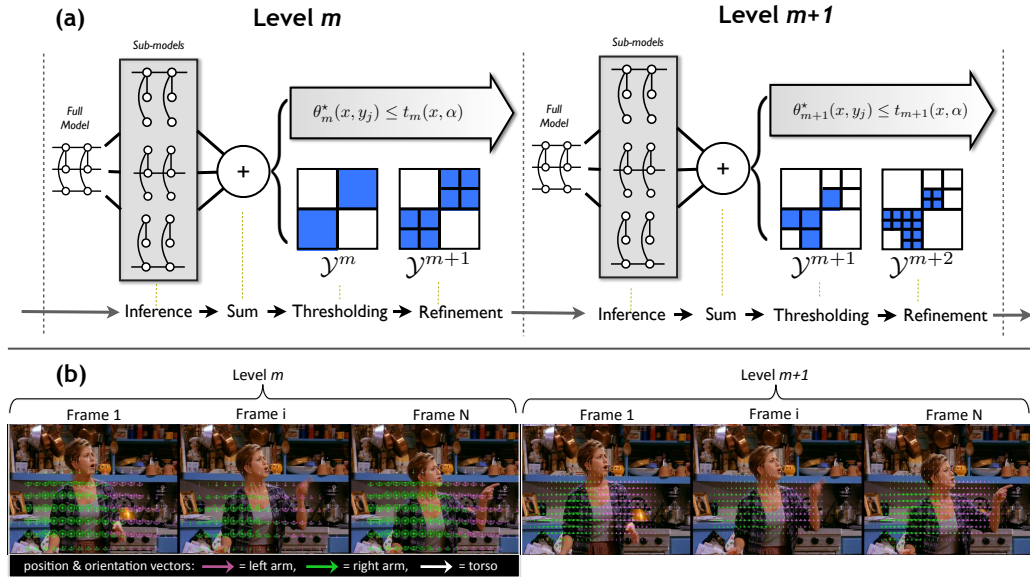

Figure 1: **(a)** Schematic overview of structured ensemble cascades. The $m$'th level of the cascade takes as input a sparse set of states $\mathcal{Y}^m$ for each variable $y_j$. The full model is decomposed into constituent sub-models (above, the three tree models used in the pose tracking experiment) and sparse inference is run. Next, the max marginals of the sub-models are summed to produce a single max marginal for each variable assignment: $\theta^\star(x, y_j) = \sum_p \theta_p^\star(x, y_j)$. Note that each level and each constituent model will have different parameters as a result of the learning process. Finally, the state spaces are thresholded based on the max-marginal scores and low-scoring states are filtered. Each state is then refined according to a state hierarchy (e.g., spatial resolution, or semantic categories) and passed to the next level of the cascade. This process can be repeated as many times as desired. In **(b)**, we illustrate two consecutive levels of the ensemble cascade on real data, showing the filtered hypotheses left for a single video example.

To maintain efficiency, inference in the sub-models of the ensemble is uncoupled (unlike in dual decomposition [5]), but the decision to filter states depends on the *sum* of the max-marginals of the constituent models (see Figure 1). We derive a convex loss function for joint estimation of sub-models in each ensemble, which provably balances accuracy and efficiency, and we propose a simple stochastic subgradient algorithm for training.

The novel contributions of this work are as follows. First, we provide a principled and practical generalization of structured cascades to intractable models. Second, we present generalization bounds on the performance of the ensemble. Third, we introduce a challenging VideoPose dataset, culled from TV videos, for evaluating pose estimation and tracking. Finally, we present an evaluation of our approach on synthetic data and the VideoPose dataset. We find that our joint training of an ensemble method outperforms several competing baselines on this difficult tracking problem.

## 2   Structured Cascades

Given an input space $\mathcal{X}$, output space $\mathcal{Y}$, and a training set $\{\langle x^1, y^1 \rangle, \ldots, \langle x^n, y^n \rangle\}$ of $n$ samples from a joint distribution $D(X, Y)$, the standard supervised learning task is to learn a hypothesis $h : \mathcal{X} \mapsto \mathcal{Y}$ that minimizes the expected loss $\mathbb{E}_D [\mathcal{L}(h(x), y)]$ for some non-negative loss function $\mathcal{L} : \mathcal{Y} \times \mathcal{Y} \to \mathbb{R}^+$. In *structured prediction problems*, $Y$ is a $\ell$-vector of variables and $\mathcal{Y} = \mathcal{Y}_1 \times \cdots \times \mathcal{Y}_\ell$, and $\mathcal{Y}_i = \{1, \ldots, K\}$. In many settings, the number of random variables, $\ell$, differs depending on input $X$, but for simplicity of notation, we assume a fixed $\ell$ here. The linear hypothesis class we consider is of the form $h(x) = \operatorname{argmax}_{y \in \mathcal{Y}} \theta(x, y)$, where the scoring function $\theta(x, y) \triangleq \theta^\top \mathbf{f}(x, y)$ is the inner product of a vector of parameters $\theta$ and a feature function $\mathbf{f} : \mathcal{X} \times \mathcal{Y} \mapsto \mathbb{R}^d$ mapping $(x, y)$ pairs to a set of $d$ real-valued features. We further assume that $\mathbf{f}$ decomposes over a set of cliques $\mathcal{C}$ over inputs and outputs, so that $\theta(x, y) = \sum_{c \in \mathcal{C}} \theta^\top \mathbf{f}_c(x, y_c)$. Above, $y_c$ is an assignment

to the subset of $Y$ variables in the clique $c$ and we will use $\mathcal{Y}_c$ to refer to the set of all assignments to the clique. By considering different cliques over $X$ and $Y$, $\mathbf{f}$ can represent arbitrary interactions between the components of $x$ and $y$. Evaluating $h(x)$ is tractable for low-treewidth (hyper)graphs but is NP-hard in general, and typically, approximate inference is used when features are not low-treewidth.

In our prior work [4], we introduced the framework of Structured Prediction Cascades (SPC) to handle problems with low-treewidth $T$ but large node state-space $K$, which makes complexity of $O(K^T)$ prohibitive. For example, for a 5-th order linear chain model for handwriting recognition or part-of-speech tagging, $K$ is about $50$, and exact inference is on the order $50^6 \approx 15$ billion times the length the sequence. In tree-structured models we have used for for human pose estimation [6], typical $K$ for each part includes image location and orientation and is on the order of $250,000$, so even $K^2$ in pairwise potentials is prohibitive. Rather than learning a single monolithic model, a structured cascade is a coarse-to-fine sequence of increasingly complex models, where model complexity scales with Markov order in sequence models or spatial/angular resolution in pose models, for example. The goal of each model is to filter out a large subset of assignments without eliminating the correct one, so that the next level only has to consider a much reduced state-space. The filtering process is feed-forward, and each stage uses inference to compute max-marginals which are used to eliminate low-scoring node or clique assignments. The parameters of each model in the cascade are learned using a loss function which balances accuracy (not eliminating correct assignment) and efficiency (eliminating as many other assignments as possible).

More precisely, for each clique assignment $y_c$, there is a *max marginal* $\theta^\star(x, y_c)$, defined as the maximum score of any output $y$ that contains the clique assignment $y_c$:

$$\theta^\star(x, y_c) \triangleq \max_{y' \in \mathcal{Y}} \ \{\theta(x, y') : y'_c = y_c\}. \tag{1}$$

For simplicity, we will examine the case where the cliques that we filter are defined only over single variables: $y_c = y_j$ (although the model may also contain larger cliques). Clique assignments are filtered by discarding any $y_j$ for which $\theta^\star(x, y_j) \leq t(x)$ for a threshold $t(x)$. We define $\mathcal{Y}_j$ to be the set of possible states for the $j$'th variable. The threshold proposed in [4] is a "max mean-max" function,

$$t(x, \alpha) = \alpha \theta^\star(x) + (1 - \alpha) \frac{1}{\sum_{j=1}^{\ell} |\mathcal{Y}_j|} \sum_{j=1}^{\ell} \sum_{y_j \in \mathcal{Y}_j} \theta^\star(x, y_j). \tag{2}$$

Filtering max marginals in this fashion can be learned because of the "safe filtering" property: ensuring that $\theta(x^i, y^i) > t(x^i, \alpha)$ is sufficient (although not necessary) to guarantee that no marginal consistent with the true answer $y^i$ will be filtered. Thus, for fixed $\alpha$, [4] proposed learning parameters $\theta$ to maximize the margin $\theta(x^i, y^i) - t(x^i, \alpha)$ and therefore minimize filtering errors:

$$\inf_{\theta, \xi \geq 0} \frac{\lambda}{2} ||\theta||^2 + \frac{1}{n} \sum_i \xi^i \quad \text{s.t.} \quad \theta(x^i, y^i) \geq t(x^i, \alpha) + \ell^i - \xi^i, \quad \forall i = 1, \dots, n \tag{3}$$

Above, $\xi^i$ are slack variables for the margin constraints, and $\ell^i$ is the size of the $i$'th example.

## 3 Structured Ensemble Cascades

In this work, we tackle the problem of learning a structured cascade for problems in which inference is *intractable*, but in which the large node state-space has a natural hierarchy that can be exploited. For example, such hierarchies arise in pose estimation by discretizing the articulation of joints at multiple resolutions, or in image segmentation due to the semantic relationship between class labels (e.g., "grass" and "tree" can be grouped as "plants," "horse" and "cow" can be grouped as "animal.")

Although the methods discussed in this section can be applied to more general intractable settings, and our prior work considered more general cascades that operate on graph cliques, we will assume for simplicity that the structured cascades operate in a "node-centric" coarse-to-fine manner as follows. For each variable $y_j$ in the model, each level of the cascade filters a current set of possible states $\mathcal{Y}_j$, and any surviving states are passed forward to the next level of the cascade by substituting each state with its set of descendents in the hierarchy. Thus, in the pose estimation problem, surviving states are subdivided into multiple finer-resolution states; in the image segmentation problem, broader object classes are split into their constituent classes for the next level.

We propose a novel method for learning structured cascades when inference is intractable due to loops in the graphical structure. The key idea of our approach is to decompose the loopy model into a collection of equivalent tractable sub-models for which inference is tractable. What distinguishes our approach from other decomposition based methods (e.g., [5, 7]) is that, because the cascade's objective is filtering and not decoding, our approach does not require enforcing the constraint that the sub-models agree on which output has maximum score. We call our approach *structured ensemble cascades*.

## 3.1 Decomposition without agreement constraints

Given a loopy (intractable) graphical model, it is always possible to express the score of a given output $\theta(x, y)$ as the sum of $P$ scores $\theta_p(x, y)$ under sub-models that collectively cover every edge in the loopy model: $\theta(x, y) = \sum_p \theta_p(x, y)$. (See Figures 2 & 3 for illustrations specific to the experiments presented in this paper.) For example, in the method of dual decomposition [5], it is possible to solve a relaxed MAP problem in the (intractable) full model by running inference in the (tractable) sub-models under the constraint that *all sub-models agree on the argmax solution.* Enforcing this constraint requires iteratively re-weighting unary potentials of the sub-models and repeatedly re-running inference until each sub-model convergences to the same argmax solution.

However, for the purposes of a structured cascade, we are only interested in computing the max marginals $\theta^\star(x, y_j)$. In other words, we are only interested in knowing whether or not a configuration $y$ consistent with $y_j$ that scores highly in each sub-model $\theta_p(x, y)$ *exists.* We show in the remainder of this section that the requirement that a *single* $y$ consistent with $y_j$ optimizes the score of each submodel (i.e, that all sub-models *agree*) is not necessary for the purposes of filtering. Thus, because we do not have to enforce agreement between sub-models, we can learn a structured cascade for intractable models, but pay only a linear (factor of $P$) increase in inference time over the tractable sub-models.

Formally, we define a single level of the ensemble cascade as a set of $P$ models such that $\theta(x, y) = \sum_p \theta_p(x, y)$. We let $\theta_p(x, \cdot)$, $\theta_p^\star(x, \cdot)$, $\theta_p^\star(x)$ and $t_p(x, \alpha)$ be the score, max marginal, max score, and threshold of the $p$'th model, respectively. We define the *argmax marginal* or *witness* $y_p^\star(x, y_j)$ to be the maximizing complete assignment of the corresponding max marginal $\theta_p^\star(x, y_j)$. Then, if $y = y_p^\star(x, y_j)$ is the same for each of the $p$'th submodels, we have that

$$\theta^\star(x, y_j) = \sum_p \theta_p^\star(x, y_j) \tag{4}$$

Note that if we do not require the sub-models to agree, then $\theta^\star(x, y_j)$ is stricly less than $\sum_p \theta_p^\star(x, y_j)$. Nonetheless, as we show next, the approximation $\theta^\star(x, y_j) \approx \sum_p \theta_p^\star(x, y_j)$ is still useful and sufficient for filtering in a structured cascade.

## 3.2 Safe filtering and generalization error

We first show that if a given label $y$ has a high score in the full model, it must also have a large ensemble max marginal score, even if the sub-models do not agree on the argmax. This results in a "safe filtering" lemma similar to that given in [4], as follows:

**Lemma 1** (Joint Safe Filtering). *If $\sum_p \theta_p(x, y) > t$, then $\sum_p \theta_p^\star(x, y_j) > t$ for all $y_j \subseteq y$.*

*Proof.* In English, this lemma states that if the global score is above a given threshold, then the sum of sub-model max-marginals is also above threshold (with no agreement constraint). The proof is straightforward. For any $y_j$ consistent with $y$, we have $\theta_p^\star(x, y_j) \geq \theta_p(x, y)$. Therefore $\sum_p \theta_p^\star(x, y_j) \geq \sum_p \theta_p(x, y) > t$. $\qquad\square$

Therefore, we see that an agreement constraint is not necessary in order to filter safely: if we ensure that the combined score $\sum_p \theta_p(x, y)$ of the true label $y$ is above threshold, then we can filter without making a mistake if we compute max marginals by running inference separately for each sub-model. However, there is still potentially a price to pay for disagreement. If the sub-models do not agree, *and* the truth is not above threshold, then the threshold may filter *all* of the states for a given variable

$y_j$ and therefore "break" the cascade. This results from the fact that without agreement, there is no single argmax output $y^\star$ that is always above threshold for any $\alpha$; therefore, it is not guaranteed that there exists an output $y$ to satisfy the Joint Safe Filtering Lemma. However, we note that in our experiments, we never experienced such breakdown of the cascades due to overly aggressive filtering.

In order to learn parameters that are useful for filtering, Lemma 1 suggests a natural *ensemble filtering loss*, which we define for any fixed $\alpha$ as follows,

$$\mathcal{L}_{joint}(\theta, \langle x, y \rangle) = \mathbf{1}\left[\sum_p \theta_p(x, y) \leq \sum_p t_p(x, \alpha)\right], \tag{5}$$

where $\theta = \{\theta_1, \dots, \theta_P\}$ is the set of all parameters of the ensemble. (Note that this loss function is somewhat conservative because it measures whether or not a *sufficient* but not necessary condition for a filtering error has occured.)

To conclude this section, we provide a generalization bound on the ensemble filtering loss, equivalent to the bounds in [4] for the single-model cascades. To do so, we first eliminate the dependence on $x$ and $\theta$ by rewriting $\mathcal{L}_{joint}$ in terms of the *scores* of every possible state assignment, $\theta \cdot \mathbf{f}(x, y_j)$, according to each sub-model. Let the vector $\theta_x \in \mathbb{R}^{mP}$ denote these scores, where $m$ is the number of possible state assignments in the sub-models.

**Theorem 1.** *For any fixed $\alpha \in [0, 1)$, define the dominating cost function $\phi(y, \theta_x) = r_\gamma(1/P \sum_p \theta_p(x, y) - t_p(x, \alpha))$, where $r_\gamma(\cdot)$ is the ramp function with slope $\gamma$. Let $||\theta_p||_2 \leq F$ for all $p$, and $||\mathbf{f}(x, y_j)||_2 \leq 1$ for all $x$ and $y_j$. Then there exists a constant $C$ such that for any integer $n$ and any $0 < \delta < 1$ with probability $1 - \delta$ over samples of size $n$, every $\theta = \{\theta_1, \dots, \theta_P\}$ satisfies:*

$$\mathbb{E}\left[\mathcal{L}_{joint}(Y, \theta_x)\right] \leq \hat{\mathbb{E}}\left[\phi(Y, \theta_x)\right] + \frac{Cm\sqrt{\ell}FP}{\gamma\sqrt{n}} + \sqrt{\frac{8\ln(2/\delta)}{n}}, \tag{6}$$

*where $\hat{\mathbb{E}}$ is the empirical expectation with respect to training data.*

The proof is given in the supplemental materials.

### 3.3 Parameter estimation with gradient descent

In this section we now discuss how to minimize the loss (5) given a dataset. We rephrase the SC optimization problem (3) using the ensemble max-marginals to form the ensemble cascade learning problem,

$$\inf_{\theta_1, \dots, \theta_P, \xi \geq 0} \frac{\lambda}{2} \sum_p ||\theta_p||^2 + \frac{1}{n} \sum_i \xi^i \quad \text{s.t.} \quad \sum_p \theta_p(x^i, y^i) \geq \sum_p t_p(x^i, \alpha) + \ell^i - \xi^i, \tag{7}$$

Seeing that the constraints can be ordered to show $\xi^i \leq \sum_p t_p(x^i, \alpha) - \sum_p \theta_p(x^i, y^i) + \ell^i$, we can form an equivalent unconstrained minimization problem and take the subgradient of (7) with respect to each parameter $\theta_p$. This yields the following update rule for the $p$'th model:

$$\theta_p \leftarrow (1 - \lambda)\theta_p + \begin{cases} 0 & \text{if } \sum_p \theta_p(x^i, y^i) \geq \sum_p t_p(x^i, \alpha) + \ell^i, \\ \nabla\theta_p(x^i, y^i) - \nabla t_p(x^i, \alpha) & \text{otherwise.} \end{cases} \tag{8}$$

This update is identical to the original SC update with the exception that we update each model individually only when the ensemble has made a mistake *jointly*. Thus, learning to filter with the ensemble requires only $P$ times as many resources as learning to filter with any of the models individually.

## 4 Experiments

We evaluated structured ensemble cascades in two experiments. First, we analyzed the "best-case" filtering performance of the summed max-marginal approximation to the true marginals on a synthetic image segmentation task, assuming the true scoring function $\theta(x, y)$ is available for inference. Second, we evaluated the real-world accuracy of our approach on a difficult, real-world human pose dataset (VideoPose). In both experiments, the max-marginal ensemble outperforms state-of-the-art baselines.

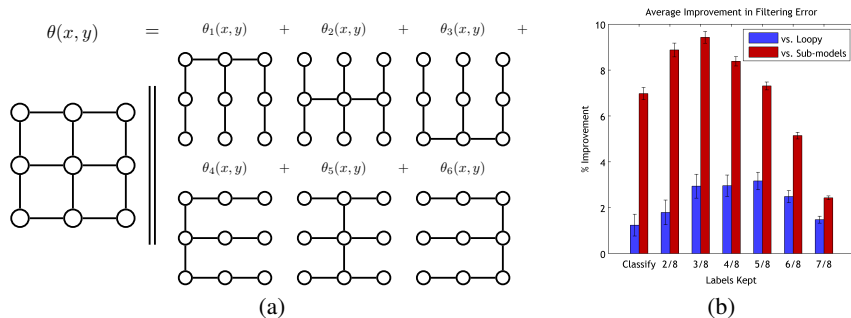

$$\theta(x,y) \quad = \quad \theta_1(x,y) \quad + \quad \theta_2(x,y) \quad + \quad \theta_3(x,y) \quad +$$

$$\theta_4(x,y) \quad + \quad \theta_5(x,y) \quad + \quad \theta_6(x,y)$$

(a)              (b)

Figure 2: (a) Example decomposition of a $3 \times 3$ fully connected grid into all six constituent "comb" trees. In general, a $n \times n$ grid yields $2n$ such trees. (b) Improvement over Loopy BP and constituent tree-models on the synthetic segmentation task. Error bars show standard error.

## 4.1 Asymptotic Filtering Accuracy

We first evaluated the filtering accuracy of the max-marginal ensemble on a synthetic 8-class segmentation task. For this experiment, we removed variability due to parameter estimation and focused our analysis on accuracy of inference. We compared our approach to Loopy Belief Propagation (Loopy BP) [8], a state-of-the-art method for approximate inference, on a $11 \times 11$ two-dimensional grid MRF.* For the ensemble, we used 22 unique "comb" tree structures to approximate the full grid model (i.e. Figure 2(a)). To generate a synthetic instance, we generated unary potentials $\omega_i(k)$ uniformly on $[0,1]$ and pairwise potentials log-uniformly: $\omega_{ij}(k,k') = \exp -v$, where $v \sim \mathcal{U}[-25, 25]$ was sampled independently for every edge and every pair of classes. (Note that for the ensemble, we normalized unary and edge potentials by dividing by the number of times that each potential was included in any model.) It is well known that inference for such grid MRFs is extremely difficult [8], and we observed that Loopy BP failed to converge for at least a few variables on most examples we generated.

**Ensemble outperforms Loopy BP.** We evaluted our approach on 100 synthetic grid MRF instances. For each instance, we computed the accuracy of filtering using marginals from Loopy BP, the ensemble, and each individual sub-model. We determined error rates by counting the number of times "ground truth" was incorrectly filtered if the top $K$ states were kept for each variable, where we sampled 1000 "ground truth" examples from the true joint distribution using Gibbs sampling. To obtain a good estimate of the true marginals, we restarted the chain for each sample and allowed 1000 iterations of mixing time. The result is presented in Figure 2(b) for all possible values of $K$ (filter aggressiveness.) We found that the ensemble outperformed Loopy BP and the individual sub-models by a significant margin for all $K$.

**Effect of sub-model agreement.** We next investigated the question of whether or not the ensembles were most accurate on variables for which the sub-models tended to agree. For each variable $y_{ij}$ in each instance, we computed the mean pairwise Spearman correlation between the ranking of the 8 classes induced by the max marginals of each of the 22 sub-models. We found that complete agreement between all sub-models never occured (the median correlation was 0.38). We found that sub-model agreement was significantly correlated ($p < 10^{-15}$) with the error of the ensemble for all values of $K$, peaking at $\rho = -0.143$ at $K = 5$. Thus, increased agreement predicted a decrease in error of the ensemble. We then asked the question: Does the effect of model agreement explain the *improvement* of the ensemble over Loopy BP? In fact, the improvement in error compared to Loopy BP was *not* correlated with sub-model agreement for any $K$ (maximum $\rho = 0.0185$, $p < 0.05$). Thus, sub-model agreement does *not* explain the improvement over Loopy BP, indicating that sub-model disagreement is not related to the difficulty in inference problems that causes Loopy BP to underperform relative to the ensembles (e.g., due to convergence failure.)

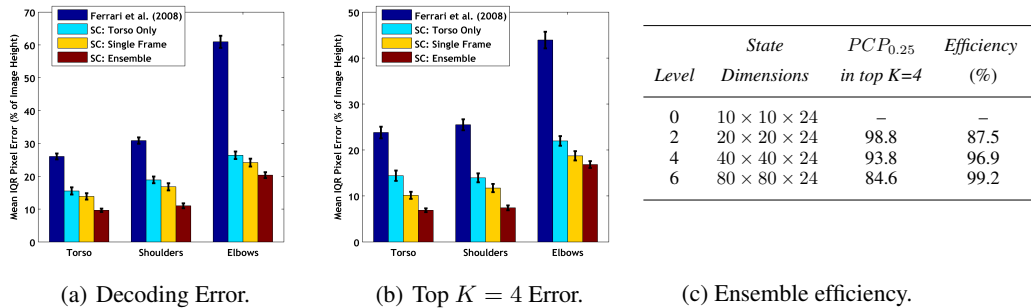

(a) Decoding Error.      (b) Top $K = 4$ Error.      (c) Ensemble efficiency.

Figure 3: **(a)**,**(b)**: Prediction error for VideoPose dataset. Reported errors are the average distance from a predicted joint location to the true joint for frames that lie in the [25,75] inter-quartile range (IQR) of errors. Error bars show standard errors computed with respect to clips. All SC models outperform [9]; the "torso only" persistence cascade introduces additional error compared to a single-frame cascade, but adding arm dependencies in the ensemble yields the best performance. **(c)**: Summary of test set filtering efficiency and accuracy for the ensemble cascade. $PCP_{0.25}$ measures Oracle % of correctly matched limb locations given unfiltered states; see [6] for more details.

## 4.2 The VideoPose Dataset

Our dataset consists of 34 video clips of approximately 50 frames each. The clips were harvested from three popular TV shows: 3 from *Buffy the Vampire Slayer*, 27 from *Friends*, and 4 from *LOST*. Clips were chosen to highlight a variety of situations and and movements when the camera is largely focused on a single actor. In our experiments, we use the *Buffy* and half of the *Friends* clips as training (17 clips), and the remaining *Friends* and *LOST* clips for testing. In total we test on 901 individual frames. The *Friends* are split so no clips from the same episode are used for both training and testing. We further set aside 4 of the *Friends* test clips to use as a development set. Each frame of each clip is hand-annotated with locations of joints of a full pose model: torso, upper/lower arms for both right and left, and top and bottom of head. For each joint, a binary tag indicating whether or not the joint is occluded is also included, to be used in future research.[†] For simplicity, we use only the torso and upper arm annotations in this work, as these have the strongest continuity across frames and strong geometric relationships.

**Articulated pose model.** All of the models we evaluated on this dataset share the same basic structure: a variable for each limb's $(x, y)$ location and angle rotation (torso, left arm, and right arm) with edges between torso and arms to model pose geometry. We refer to this basic model, evaluated independently on each frame, as the "Single Frame" approach. For the VideoPose dataset, we augmented this model by adding edges between limb states in adjacent frames (Figure 1), forming an intractable, loopy model. **Features:** Our features in a single frame are the same as in the beginning levels of the pictorial structure cascade from [6]: unary features are discretized Histogram of Gradient part detectors scores, and pairwise terms measure relative displacement in location and angle between neighboring parts. Pairwise features connecting limbs across time also express geometric displacement, allowing our model to capture the fact that human limbs move smoothly over time.

**Coarse-to-Fine Ensemble Cascade.** We learned a coarse-to-fine structured cascade with six levels for tracking as follows. The six levels use increasingly finer state spaces for joint locations, discretized into bins of resolution $10 \times 10$ up to $80 \times 80$, with each stage doubling one of the state space dimensions in the refinement step. All levels use an angular discretization of 24 bins. For the ensemble cascade, we learned three sub-models simultaneously (Figure 1), with each sub-model accounting for temporal consistency for a different limb by adding edges connecting the same limb in consecutive frames.

**Experimental Comparison.** A summary of results are presented in Figure 3. We compared the single-frame cascade and the ensemble cascade to a state-of-the-art single-frame pose detector (Ferrari et al. [9]) and to one of the individual sub-models, modeling torso consistency only ("Torso

---

[†]The VideoPose dataset is available online at `http://vision.grasp.upenn.edu/video/`.

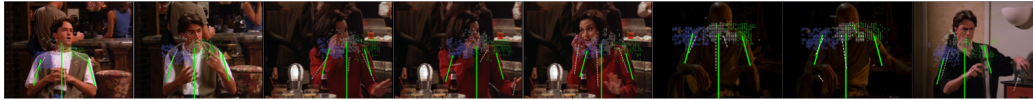

Figure 4: Qualitative test results. Points shown are the position of left/right shoulders and torsos at the last level of the ensemble SC (blue square, green dot, white circle resp.). Also shown (green line segments) are the best-fitting hypotheses to groundtruth joints, selected from within the top 4 max-marginal values. Shown as dotted gray lines is the best guess pose returned by the [9].

Only"). We evaluated the method from [9] on only the first half of the test data due to computation time (taking approximately 7 minutes/frame). We found that the ensemble cascade was the most accurate for every joint in the model, that all cascades outperformed the state-of-the-art baseline, and, interestingly, that the single-frame cascade outperformed the torso-only cascade. We suspect that the poor performance of the torso-only model may arise because propagating only torso states through time leads to an over-reliance on the relatively weak torso signal to determine the location of all the limbs. Sample qualitative output from the ensemble is presented in Figure 4.

## 5 Discussion

**Related Work.** Tracking with articulated body parts is challenging for two main reasons. First, body parts are hard to detect in unconstrained environments due to the enormous variability in appearance (from lighting, clothing and articulation) and occlusion. Second, the huge number of degrees of freedom makes exact modeling of the problem computationally prohibitive. In light of these two issues, many works focus on fixed-camera environments (e.g., [10, 11, 12]), some even assuming sillhouettes can be obtained (e.g., [2]), or 3d information from multiple sensors ([13]). In choices of modeling, past works reduce the large state space degrees of freedom by only modeling location and scale, or resorting to sampling methods ([1, 14], or embedding into low-dimensional latent spaces [10]. In contrast, in this work we learn to efficiently navigate an unconstrained state space in the challenging setting of a single, non-fixed camera.

We adopt the same basic modeling structure as [15, 9, 16] in our work, but also model dependencies through time. We also take a discriminative approach to training rather than generative. Ferrari et al. [9] use loopy belief propagation to incorporate temporal consistency of parts, but to our knowledge we are the first to quantitatively evaluate on movie/TV show sequences.

In the method of dual decomposition [5], efficient optimization of a LP relaxation of MAP inference in an intractable model is achieved by coupling the inference of a collection of tractable sub-models. This coupling is achieved by repeatedly performing inference and updating a set of dual parameters until convergence. In contrast, we perform inference independently in each sub-model only *once*, and reason about individual variables using the sums of max-marginals.

**Future Research.** Several key questions remain as future directions of research. Although we presented generalization bounds for the error of the cascade, such bounds are purely "post-hoc." We are currently investigating *a priori* properties of or assumptions about the data and cascade that will provably lead to efficient cascaded learning and inference. In the future, our approach on the VideoPose dataset could be easily extended to model more limbs, additionally complex features in time and geometry (e.g. [6]), and additional states such as occlusions. Successfully solving this problem is necessary in order to understand the context and consequences of interactions between actors in video; e.g., to be able to follow a pointing arm or to observe the transfer of an important object from one person to another.

### Acknowledgements

The authors were partially supported by NSF Grant 0803256 and ARL Cooperative Agreement W911NF-10-2-0016. David Weiss was also supported by a NSF Graduate Research Fellowship.

## Footnotes

*We used the UGM Matlab Toolbox by Mark Schmidt for the Loopy BP and Gibbs MCMC sections of this experiment. Publicly available at: http://people.cs.ubc.ca/ schmidtm/Software/UGM.html

## References

[1] L. Sigal, S. Bhatia, S. Roth, M.J. Black, and M. Isard. Tracking loose-limbed people. In *Proc. CVPR*, 2004.

[2] B. Wu and R. Nevatia. Detection and tracking of multiple, partially occluded humans by bayesian combination of edgelet based part detectors. *IJCV*, 75(2):247–266, 2007.

[3] J.D.J. Shotton, J. Winn, C. Rother, and A. Criminisi. Textonboost for image understanding: Multi-class object recognition and segmentation by jointly modeling texture, layout, and context. *IJCV*, 81(1), January 2009.

[4] D. Weiss and B. Taskar. Structured prediction cascades. In *Proc. AISTATS*, 2010.

[5] N. Komodakis, N. Paragios, and G. Tziritas. MRF optimization via dual decomposition: Message-passing revisited. In *Proc. ICCV*, 2007.

[6] B. Sapp, A. Toshev, and B. Taskar. Cascaded models for articulated pose estimation. In *Proc. ECCV*, 2010.

[7] D. P. Bertsekas. *Nonlinear Programming*. Athena Scientific, second edition, 1999.

[8] D. Koller and N. Friedman. *Probabilistic Graphical Models: Principles and Techniques*. The MIT Press, 2009.

[9] V. Ferrari, M. Marin-Jimenez, and A. Zisserman. Progressive search space reduction for human pose estimation. In *Proc. CVPR*, 2008.

[10] M. Andriluka, S. Roth, and B. Schiele. People-tracking-by-detection and people-detection-by-tracking. In *Proc. CVPR*, 2008.

[11] S. Pellegrini, A. Ess, K. Schindler, and L. Van Gool. Youll Never Walk Alone: Modeling Social Behavior for Multi-target Tracking. In *Proc. ICCV*, 2009.

[12] L. Kratz and K. Nishino. Tracking with Local Spatio-Temporal Motion Patterns in Extremely Crowded Scenes. In *Proc. CVPR*, 2010.

[13] R. Muñoz-Salinas, E. Aguirre, and M. García-Silvente. People detection and tracking using stereo vision and color. *Image and Vision Computing*, 25(6):995–1007, 2007.

[14] J. S. Kwon and K. M. Lee. Tracking of a non-rigid object via patch-based dynamic appearance modeling and adaptive basin hopping monte carlo sampling. In *Proc. CVPR*, 2009.

[15] B. Sapp, C. Jordan, and B. Taskar. Adaptive pose priors for pictorial structures. In *Proc. CVPR*, 2010.

[16] M. Andriluka, S. Roth, and B. Schiele. Pictorial structures revisited: People detection and articulated pose estimation. In *Proc. CVPR*, 2009.

